# Data-Dependent Bounds for Bayesian Mixture Methods

**Ron Meir**
Department of Electrical Engineering
Technion, Haifa 32000, Israel
rmeir@ee.technion.ac.il

**Tong Zhang**
IBM T.J. Watson Research Center
Yorktown Heights, NY 10598, USA
tzhang@watson.ibm.com

## Abstract

We consider Bayesian mixture approaches, where a predictor is constructed by forming a weighted average of hypotheses from some space of functions. While such procedures are known to lead to optimal predictors in several cases, where sufficiently accurate prior information is available, it has not been clear how they perform when some of the prior assumptions are violated. In this paper we establish data-dependent bounds for such procedures, extending previous randomized approaches such as the Gibbs algorithm to a fully Bayesian setting. The finite-sample guarantees established in this work enable the utilization of Bayesian mixture approaches in agnostic settings, where the usual assumptions of the Bayesian paradigm fail to hold. Moreover, the bounds derived can be directly applied to non-Bayesian mixture approaches such as Bagging and Boosting.

## 1 Introduction and Motivation

The standard approach to Computational Learning Theory is usually formulated within the so-called frequentist approach to Statistics. Within this paradigm one is interested in constructing an estimator, based on a finite sample, which possesses a small loss (generalization error). While many algorithms have been constructed and analyzed within this context, it is not clear how these approaches relate to standard optimality criteria within the frequentist framework. Two classic optimality criteria within the latter approach are the minimax and admissibility criteria, which characterize optimality of estimators in a rigorous and precise fashion [9]. Except in some special cases [12], it is not known whether any of the approaches used within the Learning community lead to optimality in either of the above senses of the word. On the other hand, it is known that under certain regularity conditions, Bayesian estimators lead to either minimax or admissible estimators, and thus to well-defined optimality in the classical (frequentist) sense. In fact, it can be shown that Bayes estimators are essentially the only estimators which can achieve optimality in the above senses [9]. This optimality feature provides strong motivation for the study of Bayesian approaches in a *frequentist* setting.

While Bayesian approaches have been widely studied, there have not been generally

applicable bounds in the frequentist framework. Recently, several approaches have attempted to address this problem. In this paper we establish finite sample data-dependent bounds for Bayesian mixture methods, which together with the above optimality properties suggest that these approaches should become more widely used.

Consider the problem of supervised learning where we attempt to construct an estimator based on a finite sample of pairs of examples $S = \{(\mathbf{x}_1, y_1), \ldots, (\mathbf{x}_n, y_n)\}$, each drawn independently according to an unknown distribution $\mu(\mathbf{x}, y)$. Let $\mathcal{A}$ be a learning algorithm which, based on the sample $S$, constructs a hypothesis (estimator) $h$ from some set of hypotheses $\mathcal{H}$. Denoting by $\ell(y, h(\mathbf{x}))$ the instantaneous loss of the hypothesis $h$, we wish to assess the true loss $L(h) = \mathbf{E}_\mu \ell(y, h(\mathbf{x}))$ where the expectation is taken with respect to $\mu$. In particular, the objective is to provide *data-dependent* bounds of the following form. For any $h \in \mathcal{H}$ and $\delta \in (0, 1)$, with probability at least $1 - \delta$,

$$L(h) \leq \Lambda(h, S) + \Delta(h, S, \delta), \tag{1}$$

where $\Lambda(h, S)$ is some empirical assessment of the true loss, and $\Delta(h, S, \delta)$ is a complexity term. For example, in the classic Vapnik-Chervonenkis framework, $\Lambda(h, S)$ is the empirical error $(1/n) \sum_{i=1}^n \ell(y_i, h(\mathbf{x}_i))$ and $\Delta(h, S, \delta)$ depends on the VC-dimension of $\mathcal{H}$ but is independent of both the hypothesis $h$ and the sample $S$. By algorithm and data-dependent bounds we mean bounds where the complexity term depends on both the hypothesis (chosen by the algorithm $\mathcal{A}$) and the sample $S$.

## 2   A Decision Theoretic Bayesian Framework

Consider a decision theoretic setting where we define the sample dependent loss of an algorithm $\mathcal{A}$ by $R(\mu, \mathcal{A}, S) = \mathbf{E}_\mu \ell(y, \mathcal{A}(\mathbf{x}, S))$. Let $\theta_\mu$ be the optimal predictor for $y$, namely the function minimizing $\mathbf{E}_\mu \{\ell(y, \phi(\mathbf{x}))\}$ over $\phi$. It is clear that the best algorithm $\mathcal{A}$ (Bayes algorithm) is the one that always return $\theta_\mu$, assuming $\mu$ is known. We are interested in the expected loss of an algorithm averaged over samples $S$:

$$R(\mu, \mathcal{A}) = \mathbf{E}_S R(\mu, \mathcal{A}, S) = \int R(\mu, \mathcal{A}, S) d\mu(S),$$

where the expectation is taken with respect to the sample $S$ drawn i.i.d. from the probability measure $\mu$. If we consider a family of measures $\mu$, which possesses some underlying *prior distribution* $\pi(\mu)$, then we can construct the averaged risk function with respect to the prior as,

$$r(\pi, \mathcal{A}) = \mathbf{E}_\pi R(\mu, \mathcal{A}) = \int d\mu(S) d\pi(\mu) \int R(\mu, \mathcal{A}, S) d\pi(\mu|S),$$

where $d\pi(\mu|S) = \frac{d\mu(S) d\pi(\mu)}{\int_\mu d\mu(S) d\pi(\mu)}$ is the *posterior distribution* on the $\mu$ family, which induces a posterior distribution on the sample space as $\pi_S = E_{\pi(\mu|S)} \mu$. An algorithm minimizing the Bayes risk $r(\pi, \mathcal{A})$ is referred to as a *Bayes algorithm*. In fact, for a given prior, and a given sample $S$, the optimal algorithm should return the Bayes optimal predictor with respect to the posterior measure $\pi_S$.

For many important practical problems, the optimal Bayes predictor is a linear functional of the underlying probability measure. For example, if the loss function is quadratic, namely $\ell(y, \mathcal{A}(\mathbf{x})) = (y - \mathcal{A}(\mathbf{x}))^2$, then the optimal Bayes predictor $\theta_\mu(\mathbf{x})$ is the conditional mean of $y$, namely $\mathbf{E}_\mu[y|\mathbf{x}]$. For binary classification problems, we can let the predictor be the conditional probability $\theta_\mu(\mathbf{x}) = \mu(y = 1|\mathbf{x})$ (the optimal classification decision rule then corresponds to a test of whether $\theta_\mu(\mathbf{x}) > 0.5$), which

is also a linear functional of $\mu$. Clearly if the Bayes predictor is a linear functional of the probability measure, then the optimal Bayes algorithm with respect to the prior $\pi$ is given by

$$\mathcal{A}_B(\mathbf{x}, S) = \int_\mu \theta_\mu(\mathbf{x}) d\pi(\mu|S) = \frac{\int_\mu \theta_\mu(\mathbf{x}) d\mu(S) d\pi(\mu)}{\int_\mu d\mu(S) d\pi(\mu)}. \qquad (2)$$

In this case, an optimal Bayesian algorithm can be regarded as the predictor constructed by averaging over all predictors with respect to a data-dependent posterior $\pi(\mu|S)$. We refer to such methods as Bayesian mixture methods. While the Bayes estimator $\mathcal{A}_B(\mathbf{x}, S)$ is optimal with respect to the Bayes risk $r(\pi, \mathcal{A})$, it can be shown, that under appropriate conditions (and an appropriate prior) it is also a minimax and admissible estimator [9].

In general, $\theta_\mu$ is unknown. Rather we may have some prior information about possible models for $\theta_\mu$. In view of (2) we consider a hypothesis space $\mathcal{H}$, and an algorithm based on a mixture of hypotheses $h \in \mathcal{H}$. This should be contrasted with classical approaches where an algorithm selects a single hypothesis $h$ form $\mathcal{H}$. For simplicity, we consider a countable hypothesis space $\mathcal{H} = \{h_1, h_2, \ldots\}$; the general case will be deferred to the full paper. Let $\mathbf{q} = \{q_j\}_{j=1}^\infty$ be a probability vector, namely $q_j \geq 0$ and $\sum_j q_j = 1$, and construct the composite predictor by $f_q(\mathbf{x}) = \sum_j q_j h_j(\mathbf{x})$. Observe that in general $f_q(\mathbf{x})$ may be a great deal more complex that any single hypothesis $h_j$. For example, if $h_j(\mathbf{x})$ are non-polynomial ridge functions, the composite predictor $f$ corresponds to a two-layer neural network with universal approximation power. We denote by $Q$ the probability distribution defined by $\mathbf{q}$, namely $\sum_j q_j h_j = \mathbf{E}_{h \sim Q} h$.

A main feature of this work is the establishment of data-dependent bounds on $L(\mathbf{E}_{h \sim Q} h)$, the loss of the Bayes mixture algorithm. There has been a flurry of recent activity concerning data-dependent bounds (a non-exhaustive list includes [2, 3, 5, 11, 13]). In a related vein, McAllester [7] provided a data-dependent bound for the so-called Gibbs algorithm, which selects a hypothesis at random from $\mathcal{H}$ based on the posterior distribution $\pi(h|S)$. Essentially, this result provides a bound on the average error $\mathbf{E}_{h \sim Q} L(h)$ rather than a bound on the error of the averaged hypothesis. Later, Langford *et al.* [6] extended this result to a mixture of classifiers using a margin-based loss function. A more general result can also be obtained using the covering number approach described in [14]. Finally, Herbrich and Graepel [4] showed that under certain conditions the bounds for the Gibbs classifier can be extended to a Bayesian mixture classifier. However, their bound contained an explicit dependence on the dimension (see Thm. 3 in [4]).

Although the approach pioneered by McAllester came to be known as PAC-Bayes, this term is somewhat misleading since an optimal Bayesian method (in the decision theoretic framework outline above) does not average over loss functions but rather over hypotheses. In this regard, the learning behavior of a true Bayesian method is not addressed in the PAC-Bayes analysis. In this paper, we would like to narrow the discrepancy by analyzing Bayesian mixture methods, where we consider a predictor that is the average of a family of predictors with respect to a data-dependent posterior distribution. Bayesian mixtures can often be regarded as a good approximation to a true optimal Bayesian method. In fact, we have shown above that they are equivalent for many important practical problems.

Therefore the main contribution of the present work is the extension of the above mentioned results in PAC-Bayes analysis to a rather unified setting for Bayesian mixture methods, where different regularization criteria may be incorporated, and their effect on the performance easily assessed. Furthermore, it is also essential that

the bounds obtained are *dimension-independent*, since otherwise they yield useless results when applied to kernel-based methods, which often map the input space into a space of very high dimensionality. Similar results can also be obtained using the covering number analysis in [14]. However the approach presented in the current paper, which relies on the direct computation of the Rademacher complexity, is more direct and gives better bounds. The analysis is also easier to generalize than the corresponding covering number approach. Moreover, our analysis applies directly to other non-Bayesian mixture approaches such as Bagging and Boosting.

Before moving to the derivation of our bounds, we formalize our approach. Consider a countable hypothesis space $\mathcal{H} = \{h_j\}_{j=1}^{\infty}$, and a probability distribution $\{q_j\}$ over $\mathcal{H}$. Introduce the vector notation $\sum_{k=1}^{\infty} q_k h_k(\mathbf{x}) = \mathbf{q}^\top \mathbf{h}(\mathbf{x})$. A learning algorithm within the Bayesian mixture framework uses the sample $S$ to select a distribution $Q$ over $\mathcal{H}$ and then constructs a mixture hypothesis $f_q(\mathbf{x}) = \mathbf{q}^\top \mathbf{h}(\mathbf{x})$. In order to constrain the class of mixtures used in constructing the mixture $\mathbf{q}^\top \mathbf{h}$ we impose constraints on the mixture vector $\mathbf{q}$. Let $g(\mathbf{q})$ be a non-negative convex function of $\mathbf{q}$ and define for any positive $A$,

$$\Omega_A = \{\mathbf{q} \in \mathcal{S} : g(\mathbf{q}) \leq A\} \quad ; \quad \mathcal{F}_A = \{f_q : f_q(\mathbf{x}) = \mathbf{q}^\top \mathbf{h}(\mathbf{x}) : \mathbf{q} \in \Omega_A\}, \quad (3)$$

where $\mathcal{S}$ denotes the probability simplex. In subsequent sections we will consider different choices for $g(\mathbf{q})$, which essentially acts as a regularization term. Finally, for any mixture $\mathbf{q}^\top \mathbf{h}$ we define the loss by $L(\mathbf{q}^\top \mathbf{h}) = \mathbf{E}_\mu \ell(y, (\mathbf{q}^\top \mathbf{h})(\mathbf{x}))$ and the empirical loss incurred on the sample by $\hat{L}(\mathbf{q}^\top \mathbf{h}) = (1/n) \sum_{i=1}^{n} \ell(y_i, (\mathbf{q}^\top \mathbf{h})(\mathbf{x}_i))$.

## 3   A Mixture Algorithm with an Entropic Constraint

In this section we consider an entropic constraint, which penalizes weights deviating significantly from some prior probability distribution $\boldsymbol{\nu} = \{\nu_j\}_{j=1}^{\infty}$, which may incorporate our prior information about he problem. The weights $\mathbf{q}$ themselves are chosen by the algorithm based on the data. In particular, in this section we set $g(\mathbf{q})$ to be the Kullback-Leibler divergence of $\mathbf{q}$ from $\boldsymbol{\nu}$,

$$g(\mathbf{q}) = D(\mathbf{q}\|\boldsymbol{\nu}) \quad ; \quad D(\mathbf{q}\|\boldsymbol{\nu}) = \sum_j q_j \log(q_j/\nu_j).$$

Let $\mathcal{F}$ be a class of real-valued functions, and denote by $\sigma_i$ independent Bernoulli random variables assuming the values $\pm 1$ with equal probability. We define the data-dependent Rademacher complexity of $\mathcal{F}$ as

$$\hat{R}_n(\mathcal{F}) = \mathbf{E}_{\boldsymbol{\sigma}} \left[ \sup_{f \in \mathcal{F}} \frac{1}{n} \sum_{i=1}^{n} \sigma_i f(\mathbf{x}_i) \,|\, S \right].$$

The expectation of $\hat{R}_n(\mathcal{F})$ with respect to $S$ will be denoted by $R_n(\mathcal{F})$. We note that $\hat{R}_n(\mathcal{F})$ is concentrated around its mean value $R_n(\mathcal{F})$ (e.g., Thm. 8 in [1]). We quote a slightly adapted result from [5].

**Theorem 1** (Adapted from Theorem 1 in [5])
*Let $\{\mathbf{x}_1, \mathbf{x}_2, \ldots, \mathbf{x}_n\} \in \mathcal{X}$ be a sequence of points generated independently at random according to a probability distribution $P$, and let $\mathcal{F}$ be a class of measurable functions from $\mathcal{X}$ to $\mathbb{R}$. Furthermore, let $\phi$ be a non-negative Lipschitz function with Lipschitz constant $\kappa$, such that $\phi \circ f$ is uniformly bounded by a constant $M$. Then for all $f \in \mathcal{F}$ with probability at least $1 - \delta$*

$$\mathbf{E}\phi(f(\mathbf{x})) - \frac{1}{n}\sum_{i=1}^{n} \phi(f(\mathbf{x}_i)) \leq 4\kappa R_n(\mathcal{F}) + M\sqrt{\frac{\log(1/\delta)}{2n}}.$$

An immediate consequence of Theorem 1 is the following.

**Lemma 3.1** *Let the loss function $\ell$ be bounded by $M$, and assume that it is Lipschitz with constant $\kappa$. Then for all $\mathbf{q} \in \Omega_A$ with probability at least $1 - \delta$*

$$L(\mathbf{q}^\top \mathbf{h}) \leq \hat{L}(\mathbf{q}^\top \mathbf{h}) + 4\kappa R_n(\mathcal{F}_A) + M\sqrt{\frac{\log(1/\delta)}{2n}} \ .$$

Next, we bound the empirical Rademacher average of $\mathcal{F}_A$ using $g(\mathbf{q}) = D(\mathbf{q}\|\boldsymbol{\nu})$.

**Lemma 3.2** *The empirical Rademacher complexity of $\mathcal{F}_A$ is upper bounded as follows:*

$$\hat{R}_n(\mathcal{F}_A) \leq \left(\sqrt{\frac{2A}{n}}\right) \sup_j \sqrt{\frac{1}{n}\sum_{i=1}^n h_j(\mathbf{x}_i)^2} \ .$$

**Proof:** We first recall a few facts from the theory of convex duality [10]. Let $p(\mathbf{u})$ be a convex function over a domain $U$, and set its dual $s(\mathbf{z}) = \sup_{\mathbf{u}\in U}\left(\mathbf{u}^\top \mathbf{z} - p(\mathbf{u})\right)$. It is known that $s(\mathbf{z})$ is also convex. Setting $\mathbf{u} = \mathbf{q}$ and $p(\mathbf{q}) = \sum_j q_j \log(q_j/\nu_j)$ we find that $s(\mathbf{v}) = \log\sum_j \nu_j e^{z_j}$. From the definition of $s(\mathbf{z})$ it follows that for any $\mathbf{q} \in \mathcal{S}$,

$$\mathbf{q}^\top \mathbf{z} \leq \sum_j q_j \log(q_j/\nu_j) + \log\sum_j \nu_j e^{z_j}.$$

Since $\mathbf{z}$ is arbitrary, we set $\mathbf{z} = (\lambda/n)\sum_i \sigma_i \mathbf{h}(\mathbf{x}_i)$ and conclude that for $\mathbf{q} \in \Omega_A$ and any $\lambda > 0$

$$\sup_{\mathbf{q}\in\Omega_A}\left\{\frac{1}{n}\sum_{i=1}^n \sigma_i \mathbf{q}^\top \mathbf{h}(\mathbf{x}_i)\right\} \leq \frac{1}{\lambda}\left\{A + \log\sum_j \nu_j \exp\left[\frac{\lambda}{n}\sum_i \sigma_i h_j(\mathbf{x}_i)\right]\right\} \ .$$

Taking the expectation with respect to $\sigma$, and using the Chernoff bound $\mathbf{E}_{\boldsymbol{\sigma}}\left\{\exp\left(\sum_i \sigma_i a_i\right)\right\} \leq \exp\left(\sum_i a_i^2/2\right)$, we have that

$$
\begin{aligned}
\hat{R}_n(\mathcal{F}_A) &\leq \frac{1}{\lambda}\left\{A + \mathbf{E}_{\boldsymbol{\sigma}}\log\sum_j \nu_j \exp\left[\frac{\lambda}{n}\sum_i \sigma_i h_j(\mathbf{x}_i)\right]\right\} \\
&\leq \frac{1}{\lambda}\left\{A + \sup_j \log\mathbf{E}_{\boldsymbol{\sigma}}\exp\left[\frac{\lambda}{n}\sum_i \sigma_i h_j(\mathbf{x}_i)\right]\right\} && \text{(Jensen)} \\
&\leq \frac{1}{\lambda}\left\{A + \sup_j \log\exp\left[\frac{\lambda^2}{n^2}\sum_i \frac{h_j(\mathbf{x}_i)^2}{2}\right]\right\} && \text{(Chernoff)} \\
&= \frac{A}{\lambda} + \frac{\lambda}{2n^2}\sup_j \sum_i h_j(\mathbf{x}_i)^2 \ .
\end{aligned}
$$

Minimizing the r.h.s. with respect to $\lambda$, we obtain the desired result. $\qquad\square$

Combining Lemmas 3.1 and 3.2 yields our basic bound, where $\kappa$ and $M$ are defined in Lemma 3.1.

**Theorem 2** *Let $S = \{(\mathbf{x}_1, y_1), \ldots, (\mathbf{x}_n, y_n)\}$ be a sample of i.i.d. points each drawn according to a distribution $\mu(\mathbf{x}, y)$. Let $\mathcal{H}$ be a countable hypothesis class, and set $\mathcal{F}_A$ to be the class defined in (3) with $g(\mathbf{q}) = D(\mathbf{q}\|\boldsymbol{\nu})$. Set $\Delta_{\mathcal{H}} =$*

$\left[(1/n)\mathbf{E}_\mu \sup_j \sum_{i=1}^n h_j(\mathbf{x}_i)^2\right]^{1/2}$. *Then for any* $\mathbf{q} \in \Omega_A$ *with probability at least* $1 - \delta$

$$L(\mathbf{q}^\top \mathbf{h}) \leq \hat{L}(\mathbf{q}^\top \mathbf{h}) + 4\kappa\Delta_{\mathcal{H}}\sqrt{\frac{2A}{n}} + M\sqrt{\frac{\log(1/\delta)}{2n}} \ .$$

Note that if $h_j$ are uniformly bounded, $h_j \leq c$, then $\Delta_{\mathcal{H}} \leq c$. Theorem 2 holds for a fixed value of $A$. Using the so-called multiple testing Lemma (e.g. [11]) we obtain:

**Corollary 3.1** *Let the assumptions of Theorem 2 hold, and let* $\{A_i, p_i\}$ *be a set of positive numbers such that* $\sum_i p_i = 1$. *Then for all* $A_i$ *and* $\mathbf{q} \in \Omega_{A_i}$ *with probability at least* $1 - \delta$,

$$L(\mathbf{q}^\top \mathbf{h}) \leq \hat{L}(\mathbf{q}^\top \mathbf{h}) + 4\kappa\Delta_{\mathcal{H}}\sqrt{\frac{2A_i}{n}} + M\sqrt{\frac{\log(1/p_i\delta)}{2n}} \ .$$

Note that the only distinction with Theorem 2 is the extra factor of $\log p_i$ which is the price paid for the uniformity of the bound.

Finally, we present a data-dependent bound of the form (1).

**Theorem 3** *Let the assumptions of Theorem 2 hold. Then for all* $\mathbf{q} \in \mathcal{S}$ *with probability at least* $1 - \delta$,

$$L(\mathbf{q}^\top \mathbf{h}) \leq \hat{L}(\mathbf{q}^\top \mathbf{h}) + \max(\kappa\Delta_{\mathcal{H}}, M) \times \sqrt{\frac{130D(\mathbf{q}\|\boldsymbol{\nu}) + \log(1/\delta)}{n}}. \qquad (4)$$

**Proof sketch** Pick $A_i = 2^i$ and $p_i = 1/i(i+1)$, $i = 1, 2, \ldots$ (note that $\sum_i p_i = 1$). For each $\mathbf{q}$, let $i(\mathbf{q})$ be the smallest index for which $A_{i(\mathbf{q})} \geq D(\mathbf{q}\|\boldsymbol{\nu})$ implying that $\log(1/p_{i(\mathbf{q})}) \leq 2\log\log_2(4D(\mathbf{q}\|\boldsymbol{\nu}))$. A few lines of algebra, to be presented in the full paper, yield the desired result. $\qquad\square$

The results of Theorem 3 can be compared to those derived by McAllester [8] for the randomized Gibbs procedure. In the latter case, the first term on the r.h.s. is $\mathbf{E}_{h\sim Q}\hat{L}(h)$, namely the average empirical error of the base classifiers $h$. In our case the corresponding term is $\hat{L}(\mathbf{E}_{h\sim Q}h)$, namely the empirical error of the average hypothesis. Since $\mathbf{E}_{h\sim Q}h$ is potentially much more complex than any single $h \in \mathcal{H}$, we expect that the empirical term in (4) is much smaller than the corresponding term in [8]. Moreover, the complexity term we obtain is in fact tighter than the corresponding term in [8] by a logarithmic factor in $n$ (although the logarithmic factor in [8] could probably be eliminated). We thus expect that Bayesian mixture approach advocated here leads to better performance guarantees.

Finally, we comment that Theorem 3 can be used to obtain so-called *oracle inequalities*. In particular, let $\mathbf{q}^*$ be the optimal distribution minimizing $L(\mathbf{q}^\top \mathbf{h})$, which can only be computed if the underlying distribution $\mu(\mathbf{x}, y)$ is known. Consider an algorithm which, based only on the data, selects a distribution $\hat{\mathbf{q}}$ by minimizing the r.h.s. of (4), with the implicit constants appropriately specified. Then, using standard approaches (e.g. [2]) we can obtain a bound on $L(\hat{\mathbf{q}}^\top \mathbf{h}) - L(\mathbf{q}^{*\top}\mathbf{h})$. For lack of space, we defer the derivation of the precise bound to the full paper.

# 4  General Data-Dependent Bounds for Bayesian Mixtures

The Kullback-Leibler divergence is but one way to incorporate prior information. In this section we extend the results to general *convex* regularization functions

$g(\mathbf{q})$. Some possible choices for $g(\mathbf{q})$ besides the Kullback-Leibler divergence are the standard $L_p$ norms $\|\mathbf{q}\|_p$.

In order to proceed along the lines of Section 3, we let $s(\mathbf{z})$ be the convex function associated with $g(\mathbf{q})$, namely $s(\mathbf{z}) = \sup_{\mathbf{q}\in\Omega_A}\{\mathbf{q}^\top\mathbf{z} - g(\mathbf{q})\}$. Repeating the arguments of Section 3 we have for any $\lambda > 0$ that $\frac{1}{n}\sum_{i=1}^n \sigma_i \mathbf{q}^\top \mathbf{h}(\mathbf{x}_i) \leq \frac{1}{\lambda}\left\{A + s\left(\frac{\lambda}{n}\sum_i \sigma_i\mathbf{h}(\mathbf{x}_i)\right)\right\}$, which implies that

$$\hat{R}_n(\mathcal{F}_A) \leq \inf_{\lambda\geq 0}\frac{1}{\lambda}\left\{A + \mathbf{E}_{\boldsymbol{\sigma}}s\left(\frac{\lambda}{n}\sum_i \sigma_i\mathbf{h}(\mathbf{x}_i)\right)\right\}. \tag{5}$$

Assume that $s(\mathbf{z})$ is second order differentiable, and that for any $\mathbf{h} = \sum_{i=1}^n \sigma_i \mathbf{h}(\mathbf{x}_i)$ $\frac{1}{2}(s(\mathbf{h}+\Delta\mathbf{h}) + s(\mathbf{h}-\Delta\mathbf{h})) - s(\mathbf{h}) \leq u(\Delta\mathbf{h})$. Then, assuming that $s(\mathbf{0}) = 0$, it is easy to show by induction that

$$\mathbf{E}_{\boldsymbol{\sigma}}s\left((\lambda/n)\sum_{i=1}^n \sigma_i\mathbf{h}(\mathbf{x}_i)\right) \leq \sum_{i=1}^n u((\lambda/n)\mathbf{h}(\mathbf{x}_i)). \tag{6}$$

In the remainder of the section we focus on the the case of regularization based on the $L_p$ norm. Consider $p$ and $q$ such that $1/q + 1/p = 1$, $p \in (1,\infty)$, and let $p' = \max(p,2)$ and $q' = \min(q,2)$. Note that if $p \leq 2$ then $q \geq 2, q' = p' = 2$ and if $p > 2$ then $q < 2, q' = q, p' = p$. Consider $p$-norm regularization $g(\mathbf{q}) = \frac{1}{p'}\|\mathbf{q}\|_p^{p'}$, in which case $s(\mathbf{z}) = \frac{1}{q'}\|\mathbf{z}\|_q^{q'}$. The Rademacher averaging result for $p$-norm regularization is known in the Geometric theory of Banach spaces (type structure of the Banach space), and it also follows from Khinchtine's inequality. We show that it can be easily obtained in our framework.

In this case, it is easy to see that $s(\mathbf{z}) = \frac{1}{q'}\|\mathbf{z}\|_q^{q'}$ implies $u(\mathbf{h}(\mathbf{x})) \leq \frac{q-1}{q'}\|\mathbf{h}(\mathbf{x})\|_q^{q'}$. Substituting in (5) we have

$$\hat{R}_n(\mathcal{F}_A) \leq \inf_{\lambda\geq 0}\frac{1}{\lambda}\left\{A + \frac{q-1}{q'}\left(\frac{\lambda}{n}\right)^{q'}\sum_{i=1}^n\|\mathbf{h}(\mathbf{x}_i)\|_q^{q'}\right\} = \frac{C_q}{n^{1/p'}}A^{1/p'}\left(\frac{1}{n}\sum_{i=1}^n\|\mathbf{h}(\mathbf{x}_i)\|_q^{q'}\right)^{1/q'}$$

where $C_q = ((q-1)/q')^{1/q'}$.

Combining this result with the methods described in Section 3, we establish a bound for regularization based on the $L_p$ norm. Assume that $\|\mathbf{h}(\mathbf{x}_i)\|_q$ is finite for all $i$, and set $\Delta_{\mathcal{H},q} = \left(\mathbf{E}\left\{(1/n)\sum_{i=1}^n\|\mathbf{h}(\mathbf{x}_i)\|_q^{q'}\right\}\right)^{1/q'}$.

**Theorem 4** *Let the conditions of Theorem 3 hold and set* $g(\mathbf{q}) = \frac{1}{p'}\|\mathbf{q}\|_p^{p'}$, $p \in (1,\infty)$. *Then for all* $\mathbf{q} \in \mathcal{S}$, *with probability at least* $1 - \delta$,

$$L(\mathbf{q}^\top\mathbf{h}) \leq \hat{L}(\mathbf{q}^\top\mathbf{h}) + \max(\kappa\Delta_{\mathcal{H},q}, M) \times O\left(\frac{\|\mathbf{q}\|_p}{n^{1/p'}} + \sqrt{\frac{\log\log(\|\mathbf{q}\|_p + 3) + \log(1/\delta)}{n}}\right)$$

*where* $O(\cdot)$ *hides a universal constant that depends only on* $p$.

## 5 Discussion

We have introduced and analyzed a class of regularized Bayesian mixture approaches, which construct complex composite estimators by combining hypotheses

from some underlying hypothesis class using data-dependent weights. Such weighted averaging approaches have been used extensively within the Bayesian framework, as well as in more recent approaches such as Bagging and Boosting. While Bayesian methods are known, under favorable conditions, to lead to optimal estimators in a frequentist setting, their performance in agnostic settings, where no reliable assumptions can be made concerning the data generating mechanism, has not been well understood. Our data-dependent bounds allow the utilization of Bayesian mixture models in general settings, while at the same time taking advantage of the benefits of the Bayesian approach in terms of incorporation of prior knowledge. The bounds established, being independent of the cardinality of the underlying hypothesis space, can be directly applied to kernel based methods.

**Acknowledgments** We thank Shimon Benjo for helpful discussions. The research of R.M. is partially supported by the fund for promotion of research at the Technion and by the Ollendorff foundation of the Electrical Engineering department at the Technion.

# References

[1] P. Bartlett and S. Mendelson. Rademacher and Gaussian complexities: risk bounds and structural results. In *Proceedings of the Fourteenth Annual Conference on Computational Learning Theory*, pages 224–240, 2001.

[2] P.L. Bartlett, S. Boucheron, and G. Lugosi. Model selection and error estimation. *Machine Learning*, 48:85–113, 2002.

[3] O. Bousquet and A. Chapelle. Stability and generalization. *J. Machine Learning Research*, 2:499–526, 2002.

[4] R. Herbrich and T. Graepel. A PAC-bayesian margin bound for linear classifiers; why svms work. In *Advances in Neural Information Processing Systems 13*, pages 224–230, Cambridge, MA, 2001. MIT Press.

[5] V. Koltchinksii and D. Panchenko. Empirical margin distributions and bounding the generalization error of combined classifiers. *Ann. Statis.*, 30(1), 2002.

[6] J. Langford, M. Seeger, and N. Megiddo. An improved predictive accuracy bound for averaging classifiers. In *Proceeding of the Eighteenth International Conference on Machine Learning*, pages 290–297, 2001.

[7] D. A. McAllester. Some PAC-bayesian theorems. In *Proceedings of the eleventh Annual conference on Computational learning theory*, pages 230–234, New York, 1998. ACM Press.

[8] D. A. McAllester. PAC-bayesian model averaging. In *Proceedings of the twelfth Annual conference on Computational learning theory*, New York, 1999. ACM Press.

[9] C. P. Robert. *The Bayesian Choice: A Decision Theoretic Motivation.* Springer Verlag, New York, 1994.

[10] R.T. Rockafellar. *Convex Analysis.* Princeton University Press, Princeton, N.J., 1970.

[11] J. Shawe-Taylor, P. Bartlett, R.C. Williamson, and M. Anthony. Structural risk minimization over data-dependent hierarchies. *IEEE trans. Inf. Theory*, 44:1926–1940, 1998.

[12] Y. Yang. Minimax nonparametric classification - part I: rates of convergence. *IEEE Trans. Inf. Theory*, 45(7):2271–2284, 1999.

[13] T. Zhang. Generalization performance of some learning problems in hilbert functional space. In *Advances in Neural Information Processing Systems 15*, Cambridge, MA, 2001. MIT Press.

[14] T. Zhang. Covering number bounds of certain regularized linear function classes. *Journal of Machine Learning Research*, 2:527–550, 2002.
